# Discriminative Network Models of Schizophrenia

**Guillermo A. Cecchi, Irina Rish**
IBM T. J. Watson Research Center
Yorktown Heights, NY, USA

**Benjamin Thyreau**
Neurospin
CEA, Saclay, France

**Bertrand Thirion**
INRIA
Saclay, France

**Marion Plaze**
INSERM - CEA - Univ. Paris Sud
Research Unit U.797
Neuroimaging & Psychiatry
SHFJ & Neurospin, Orsay, France

**Marie-Laure Paillere-Martinot**
AP-HP, Adolescent Psychopathology
and Medicine Dept., Maison de Solenn,
Cochin Hospital, University Paris Descartes
F-75014 Paris, France

**Catherine Martelli**
Departement de Psychiatrie
et d'Addictologie
Centre Hospitalier Paul Brousse
Villejuif, France

**Jean-Luc Martinot**
INSERM - CEA - Univ. Paris Sud
Research Unit U.797
Neuroimaging & Psychiatry
SHFJ & Neurospin, Orsay, France

**Jean-Baptiste Poline**
Neurospin
CEA, Saclay, France

## Abstract

Schizophrenia is a complex psychiatric disorder that has eluded a characterization in terms of local abnormalities of brain activity, and is hypothesized to affect the collective, "emergent" working of the brain. We propose a novel data-driven approach to capture emergent features using *functional brain networks* [4] extracted from fMRI data, and demonstrate its advantage over traditional region-of-interest (ROI) and local, task-specific linear activation analyzes. Our results suggest that schizophrenia is indeed associated with disruption of global brain properties related to its functioning as a network, which cannot be explained by alteration of local activation patterns. Moreover, further exploitation of interactions by sparse Markov Random Field classifiers shows clear gain over linear methods, such as Gaussian Naive Bayes and SVM, allowing to reach 86% accuracy (over 50% baseline - random guess), which is quite remarkable given that it is based on a single fMRI experiment using a simple auditory task.

## 1 Introduction

It has been long recognized that extracting an informative set of application-specific features from the raw data is essential in practical applications of machine learning, and often contributes even more to the success of learning than the choice of a particular classifier. In biological applications, such as brain image analysis, proper feature extraction is particularly important since the primary objective of such studies is to gain a scientific insight rather than to learn a "black-box" predictor; thus, the focus shifts towards the discovery of predictive patterns, or "biomarkers", forming a basis for *interpretable* predictive models. Conversely, biological knowledge can drive the definition of features and lead to more powerful classification.

The objective of this work is to identify biomarkers predictive of schizophrenia based on fMRI data collected for both schizophrenic and non-schizophrenic subjects performing a simple auditory task in the scanner [14]. Unlike some other brain disorders (e.g., stroke or Parkinsons disease), schizophrenia appears to be "delocalized", i.e. difficult to attribute to a dysfunction of some par-

ticular brain areas[1]. The failure to identify specific areas, as well as the controversy over which localized mechanisms are responsible for the symptoms associated with schizophrenia, have led us amongst others [7, 1, 10] to hypothesize that this disease may be better understood as a disruption of the emergent, collective properties of normal brain states, which can be better captured by *functional networks* [4], based on inter-voxel correlation strength, as opposed (or limited) to activation failures localized to specific, task-dependent areas.

To test this hypothesis, we measured diverse topological features of the functional networks and compared them across the normal subjects and schizophrenic patients groups. Specifically, we decided to ask the following questions: (1) What specific effects does schizophrenia have on the functional connectivity of brain networks? (2) Does schizophrenia affect functional connectivity in ways that are congruent with the effect it has on area-specific, task-dependent activations? (3) Is it possible to use functional connectivity to improve the classification accuracy of schizophrenic patients?

In answer to these questions, we will show that degree maps, which assign to each voxel the number of its neighbors in a network, identify spatially clustered groups of voxels with statistically significant group (i.e. normal vs. schizophrenic) differences; moreover, these highly significant voxel subsets are quite stable over different data subsets. In contrast, standard linear activation maps commonly used in fMRI analysis show much weaker group differences as well as stability. Moreover, degree maps yield very informative features, allowing for up to 86% classification accuracy (with 50% baseline), as opposed to standard local voxel activations. The best accuracy is achieved by further exploiting non-local interactions with probabilistic graphical models such as Markov Random Fields, as opposed to linear classifiers.

Finally, we demonstrate that traditional approaches based on a direct comparison of the correlation at the level of relevant regions of interest (ROIs) or using a functional parcellation technique [17], do not reveal any statistically significant differences between the groups. Indeed, a more data-driven approach that exploits properties of voxel-level networks appears to be necessary in order to achieve high discriminative power.

## 2 Background and Related Work

In Functional Magnetic Resonance Imaging (fMRI), a MR scanner non-invasively records a subject's blood-oxygenation-level dependent (BOLD) signal, known to be correlated with neural activity, as a subject performs a task of interest (e.g., viewing a picture or reading a sentence). Such scans produce a sequence of 3D images, where each image typically has on the order of 10,000-100,000 subvolumes, or *voxels*, and the sequence typically contains a few hundreds of time points, or TRs (time repetitions). Standard fMRI analysis approaches, such as the General Linear Model (GLM) [9], examine *mass-univariate* relationships between each voxel and the stimulus in order to build so-called *statistical parametric maps* that associate each voxel with some statistics that reflects its relationship to the stimulus. Commonly used *activation maps* depict the "activity" level of each voxel determined by the linear correlation of its time course with the stimulus (see Supplemental Material for details).

Clearly, such univariate analysis can miss important information contained in the interactions among voxels. Indeed, as it was shown in [8], highly predictive models of mental states can be built from voxels with sub-maximal activation. Recently, applying multivariate predictive methods to fMRI became an active area of research, focused on predicting "mental states" from fMRI data [11, 13, 2]. However, our focus herein is not just predictive modeling, but rather discovery of interpretable features with high discriminative power. Also, our problem is much more high-dimensional, since each sample (e.g., schizophrenic vs. non-schizophrenic) corresponds to a sequence of 3D images over about 400 time points, rather than to a single 3D image as in [11, 13, 2].

While the importance of modeling brain connectivity and interactions became widely recognized in the current fMRI-analysis literature [6, 19, 16], practical applications of the proposed approaches such as dynamic causal modeling [6], dynamic Bays nets [19], or structural equations [16] were

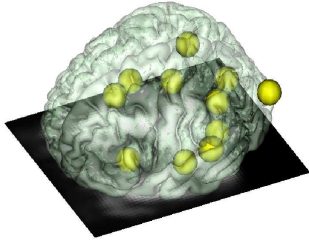

| | ROI name | (x,y,z) position | Anatomical position |
|---|---|---|---|
| 1 | 'Temporal_mid_L' | -44,-48,4 | Left temporal |
| 2 | 'Temporal_mid_et_sup_L' | -56,-36,0 | Middle and superior left temporal |
| 3 | 'Frontal_inf_L' | -40,28,0 | Left Inferior frontal |
| 4 | 'cuneus_L' | -12,-72,24 | Left cuneus |
| 5 | 'Temporal_sup_et_mid_L' | -52,-16,-8 | Middle and superior left temporal |
| 6 | 'Angular_L' | -44,-48,32 | Left angular gyrus |
| 7 | 'Temporal_sup_R' | 40,-64,24 | Right superior temporal |
| 8 | 'Angular_R' | 40,-64,24 | Right angular gyrus |
| 9 | 'Cingulum_post_R' | 4,-32,24 | Right posterior cingulum |
| 10 | 'ACC' | 0,20,30 | Anterior cingulated cortex |

Figure 1: Regions of Interest and their location on standard brain.

usually limited to interactions analysis among just a few (e.g., less than 15) known brain regions believed to be relevant to the task or phenomenon of interest. In this paper, we demonstrate that such model-based region-of-interest (ROI) analysis may fail to reveal informative interactions which, nevertheless, become visible at the finer-grain voxel level when using a purely data-driven, network-based approach [4]. Moreover, while recent publications have already indicated that functional networks in the schizophrenic brain display disrupted topological properties, we demonstrate, for the first time, that (1) specific topological properties (e.g. voxel degrees) of functional networks can help to construct highly-predictive schizophrenia classifiers that generalize well and (2) functional network differences cannot be attributed to alteration of local activation patterns, a hypothesis that was not ruled out by the results of [1, 10] and similar work.

## 3   Experimental Setup

The present study is a reanalysis of image datasets previously acquired according to the methodology described in [14]. Two groups of 12 subjects each were submitted to the same experimental paradigm involving language: schizophrenic patients and age-matched normal controls (same experiment was performed with a third group of alcoholic patients, yielding similar results - see Suppl. Materials for details). The studies had been performed after approval of the local ethics committee and all subjects were studied after they gave written informed consent. The task is based on auditory stimuli; subjects listen to emotionally neutral sentences either in native (French) or foreign language. Average length (3.5 sec mean) or pitch of both kinds of sentences is normalized. In order to catch attention of subjects, each trial begins with a short (200 ms) auditory tone, followed by the actual sentence. The subject's attention is asserted through a simple validation task: after each played sentences, a short pause of 750 ms is followed by a 500 ms two-syllable auditory cue, which belongs to the previous sentence or not, to which the subject must answer to by yes (the cue is part of the previous sentence) or no with push-buttons, when the language of the sentence was his own. For each subject, two fMRI acquisition runs are acquired, each of which consisted of 420-scans (from which the first 4 are discarded to eliminate T1 effect). A full fMRI run contains 96 trials, with 32 sentences in French (native), 32 sentences in foreign languages, and 32 silence interval controls. Data were spatially realigned and warped into the MNI template and smoothed (FWHM of 5mm) using SPM5 (www.fil.ucl.ac.uk); also, standard SPM5 motion correction was performed. Several subjects were excluded from the consideration due to excessive head motion in the scanner, leaving us with 11 schizophrenic and 11 healthy subjects, i.e. the total of 44 samples (there were two samples per subject, corresponding to the two runs of the experiment). Each sample associated with roughly 53,000 voxels (after removing out-of-brain voxels from the original $53 \times 63 \times 46$ image), over 420 time points (TRs), i.e. with more than 22,000,000 voxels/variables. Thus, some kind of dimensionality reduction and/or feature extraction is necessary prior to learning a predictive model.

## 4   Methods

We explored two different data analysis approaches aimed at discovery of discriminative patterns: (1) model-driven approaches based on prior knowledge about the regions of interest (ROI) that are believed to be relevant to schizophrenia, or model-based functional clustering, and (2) data-driven approaches based on various features extracted from the fMRI data, such as standard activation maps and a set of topological features derived from functional networks.

### 4.1   Model-Driven Approach using ROI

First, we decided to test whether the interactions between several known regions of interest (ROIs) would contain enough discriminative information about schizophrenic versus normal subjects. Ten

regions of interests (ROI) were defined using previous literature on schizophrenia and language studies, including inferior, middle and superior left temporal cortex, left inferior temporal cortex, left cuneus, left angular gyrus, right superior temporal, right angular gyrus, right posterior cingulum, and anterior cingular cortex (Figure 1). Each region was defined as a sphere of 12mm diameter centered on the x,y,z coordinates of the corresponding ROI. Because predefined regions of interest may be based on too much a priori knowledge and miss important areas, we also ran a more exploratory analysis. A second set of 600 ROI's was defined automatically using a parcellation algorithm [17] that estimates, for each subject, a collection of regions based on task-based functional signal similarity and position in the MNI space.

Time series were extracted as the spatial mean over each ROI, leading to 10 time series per subject for the predefined ROIs and 600 for the parcellation technique. The connectivity measures were of two kinds. First, the correlation coefficient was computed along time between ROIs blindly with respect to the experimental paradigm. Additionally, we computed a psycho-physiological interaction (PPI), by contrasting the correlation coefficient weighted by experimental conditions (i.e. correlation weighted by the "Language French" condition versus correlation weighted by "Control" condition after convolution with a standard hemodynamic response function). Those connectivity measures were then tested for significance using standards non parametric tests between groups (Wilcoxon signed-rank test) with corrected p-values for multiple comparisons.

## 4.2 Data-driven Approach: Feature Extraction

**Topological Features and Degree Maps.** In order to continue investigating possible disruptions of global brain functioning associated with schizophrenia, we decided to explore lower-level (as compared to ROI-level) *functional* brain networks [4] constructed at the voxel level: (1) pair-wise Pearson correlation coefficients are computed among all pairs of time-series $(v_i(t), v_j(t))$ where $v_i(i)$ corresponds to the BOLD signal of $i$-th voxel; (2) an edge between a pair of voxels $(i, j)$ is included in the network if the correlation between $v_i$ and $v_j$ exceeds a specified threshold (herein, we used the same threshold of c(Pearson)=0.7 for all voxel pairs).

For each subject, and each run, a separate functional network was constructed. Next, we measured a number of its topological features, including the *degree distribution*, *mean degree*, the size of the largest connected subgraph (*giant component*), and so on (see the supplemental material for the full list). Besides global topological features, we also computed a series of *degree maps* based on the individual voxel degree in functional network: (1) *full degree maps*, where the value assigned to each voxel is the total number of links in the corresponding network node, (2) *long-distance degree maps*, where the value is the number of links making non-local connections (5 voxels apart or more), and (3) *inter-hemispheric degree maps*, where only links reaching across the brain hemispheres are considered when computing each voxel's degree.

**Activation maps.** To find out whether local task-dependent linear activations alone could possibly explain the differences between the schizophrenic and normal brains, we used as a baseline set of features based on the standard voxel activation maps. For each subject, and for each run, activation maps, as well as their differences, or activation contrast maps, were obtained using several regressors based on the language task, as described in the supplemental material (for simplicity, we will refer to all such maps as activation maps). The activation values of each voxel were subsequently used as features in the classification task. Similarly to degree maps, we also computed a global feature, mean-activation (*mean-t-val)*), by taking the mean absolute value of the voxel's t-statistics. Both activation and degree maps for each sample were also normalized, i.e. divided by their maximal value for the given sample.

## 4.3 Classification Approaches

First, off-the-shelf methods such Gaussian Naive Bayes (GNB) and Support Vector Machines (SVM) were used in order to compare the discriminative power of different sets of features described above. Moreover, we decided to further investigate our hypothesis that interactions among voxels contain highly discriminative information, and compare those linear classifiers against probabilistic graphical models that explicitly model such interactions. Specifically, we learn a classifier based on a sparse Gaussian Markov Random Field (MRF) model [12], which leads to a convex problem with unique optimal solution, and can be solved efficiently; herein, we used the COVSEL procedure [12]. The weight on the $l_1$-regularization penalty serves as a tuning parameter of the classifier, allowing to control the sparsity of the model, as described below.

**Sparse Gaussian MRF classifier.** Let $X = \{X_1, ..., X_p\}$ be a set of $p$ random variables (e.g., voxels), and let $G = (V, E)$ be an undirected graphical model (Markov Network, or MRF) representing conditional independence structure of the joint distribution $P(X)$. The set of vertices $V = \{1, ..., p\}$ is in the one-to-one correspondence with the set $X$. There is no edge between $X_i$ and $X_j$ if and only if the two variables are conditionally independent given all remaining variables. Let $\mathbf{x} = (x_1, ..., x_p)$ denote a random assignment to $X$. We will assume a multivariate Gaussian probability density $p(\mathbf{x}) = (2\pi)^{-p/2} \det(C)^{\frac{1}{2}} e^{-\frac{1}{2}\mathbf{x}^T C \mathbf{x}}$, where $C = \Sigma^{-1}$ is the inverse covariance matrix, and the variables are normalized to have zero mean. Let $\mathbf{x}_1, ..., \mathbf{x}_n$ be a set of $n$ i.i.d. samples from this distribution, and let $S = \frac{1}{n}\sum_{i=1}^{n} \mathbf{x}_i^T \mathbf{x}_i$ denote the empirical covariance matrix. Missing edges in the above graphical model correspond to zero entries in the inverse covariance matrix $C$, and thus the problem of learning the structure for the above probabilistic graphical model is equivalent to the problem of learning the zero-pattern of the inverse-covariance matrix [2]. A popular approach is to use $l_1$-norm regularization that is known to promote sparse solutions, while still allowing (unlike non-convex $l_q$-norm regularization with $0 < q < 1$) for efficient optimization. From the Bayesian point of view, this is equivalent to assuming that the parameters of the inverse covariance matrix $C = \Sigma^{-1}$ are independent random variables $C_{ij}$ following the Laplace distributions $p(C_{ij}) = \frac{\lambda_{ij}}{2} e^{-\lambda_{ij}|C_{ij} - \alpha_{ij}|}$ with zero *location parameters* (means) $\alpha_{ij}$ and equal *scale parameters* $\lambda_{ij} = \lambda$. Then $p(C) = \prod_{i=1}^{p} \prod_{j=1}^{p} p(C_{ij}) = (\lambda/2)^{p^2} e^{-\lambda||C||_1}$, where $||C||_1 = \sum_{ij} |C_{ij}|$ is the (vector) $l_1$-norm of $C$. Assume a fixed parameter $\lambda$, our objective is to find $\arg\max_{C \succ 0} p(C|\mathbf{X})$, where $\mathbf{X}$ is the $n \times p$ data matrix, or equivalently, since $p(C|\mathbf{X}) = P(\mathbf{X}, C)/p(\mathbf{X})$ and $p(\mathbf{X})$ does not include $C$, to find $\arg\max_{C \succ 0} P(\mathbf{X}, C)$, over positive definite matrices $C$. This yields the following optimization problem considered, for example, in [12]

$$\max_{C \succ 0} \ln \det(C) - \operatorname{tr}(SC) - \lambda||C||_1$$

where $\det(A)$ and $\operatorname{tr}(A)$ denote the determinant and the trace (the sum of the diagonal elements) of a matrix $A$, respectively. For the classification task, we estimate on the training data the Gaussian conditional density $p(\mathbf{x}|y)$ (i.e. the (inverse) covariance matrix parameter) for each class $Y = \{0, 1\}$ (schizophrenic vs non-schizophrenic), and then choose the most-likely class label $\arg\max_c p(\mathbf{x}|c)P(c)$ for each unlabeled test sample $\mathbf{x}$.

**Variable Selection**: We used variable selection as a preprocessing step before applying a particular classifier, in order to (1) reduce the computational complexity of classification (especially for sparse MRF, which, unlike GNB and SVM, could not be directly applied to over 50,000 variables), (2) reduce noise and (3) identify relatively small predictive subsets of voxels. We applied a simple filter-based approach, selecting a subset of top-ranked voxels, where the ranking criterion used p-values resulting from the paired t-test, with the null-hypothesis being that the voxel values corresponding to schizophrenic and non-schizophrenic subjects came from distributions with equal means. The variables were ranked in the ascending order of their p-values (lower $p$ = higher confidence in between-group differences), and classification results on top $k$ voxels will be presented for a range of $k$ values.

**Evaluation via Cross-validation**. We used *leave-one-subject-out* rather than leave-one-sample-out cross-validation, since the two runs (two samples) for each subject are clearly not i.i.d. and must be handled together to avoid biases towards overly-optimistic results.

# 5   Results

**Model-driven ROI analysis.** First, we observed that correlations (blind to experimental paradigm) between regions and *within subjects* were very strong and significant (p-value of 0.05, corrected for the number of comparisons) when tested against 0 for all subjects (mean correlation $> 0.8$ for every group). However, these inter-region correlations do not seem to differ significantly between the groups. The parcellation technique led to some smaller p-values, but also to a stricter correction for multiple comparison and no correlation was close to the corrected threshold. Concerning the psycho-physiological interaction, results were closer to significance, but did not survive multiple comparisons. In conclusion, we could not detect significant differences between the schizophrenic patient data and normal subjects in either the BOLD signal correlation or the interaction between the signal and the main experimental contrast (native language versus silence).

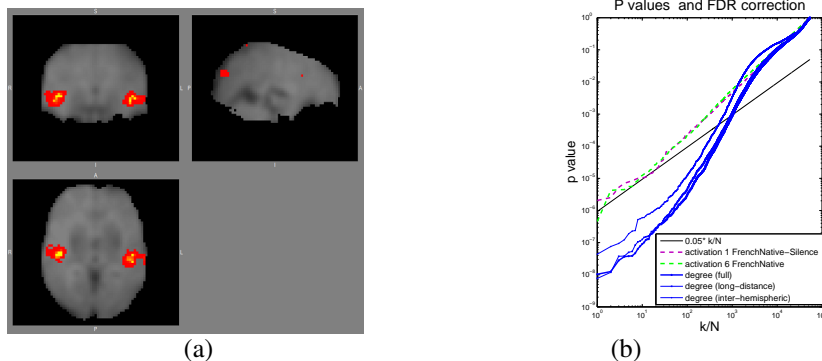

| (a) | (b) |

Figure 2: (a) FDR-corrected 2-sample t-test results for (normalized) degree maps, where the null hypothesis at each voxel assumes no difference between the schizophrenic vs normal groups. Red/yellow denotes the areas of low p-values passing FDR correction at $\alpha = 0.05$ level (i.e., 5% false-positive rate). Note that the mean (normalized) degree at those voxels was always (significantly) *higher* for normals than for schizophrenics. (b) Direct comparison of voxel p-values and FDR threshold: p-values sorted in ascending order; FDR test select voxels with $p < \alpha \cdot k/N$ ($\alpha$ - false-positive rate, $k$ - the index of a p-value in the sorted sequence, $N$ - the total number of voxels). Degree maps yield a large number (1033, 924 and 508 voxels in full, long-distance and inter-hemispheric degree maps, respectively) of highly-significant (very low) p-values, staying far below the FDR cut-off line, while only a few voxels survive FDR in case of activation maps: 7 and 2 voxels in activation maps 1 (contrast "FrenchNative - Silence") and 6 ("FrenchNative"), respectively (the rest of the activation maps do not survive the FDR correction at all).

**Data-driven analysis: topological vs activation features.** Empirical results are consistent with our hypothesis that schizophrenia disrupts the normal structure of functional networks in a way that is not derived from alterations in the activation; moreover, they demonstrate that topological properties are highly predictive, consistently outperforming predictions based on activations.

**1. Voxel-wise statistical analysis.** Degree maps show much *stronger statistical differences* between the schizophrenic vs. non-schizophrenic groups than the activation maps. Figure 2 show the 2-sample t-test results for the full degree map and the activation maps, after False-Discovery Rate (FDR) correction for multiple comparisons (standard in fMRI analysis), at $\alpha = 0.05$ level (i.e., 5% false-positive rate). While the degree map (Figure 2a) shows statistically significant differences bilaterally in auditory areas (specifically, normal group has *higher degrees* than schizophrenic group), the activation maps show almost no significant differences at all: practically no voxels there survived the FDR correction (Figure 2b. This suggests that (a) the differences in the collective behavior cannot be explained by differences in the linear task-related response, and that (b) topology of voxel-interaction networks is more informative than task-related activations, suggesting an abnormal degree distribution for schizophrenic patients that appear to lack hubs in auditory cortex, i.e., have significantly lower (normalized) voxel degrees in that area than the normal group (possibly due to a more even spread of degrees in schizophrenic vs. normal networks). Moreover, degree maps demonstrate much *higher stability* than activation maps with respect to selecting a subset of top ranked voxels over different subsets of data. Figure 3a shows that degree maps have up to almost 70% top-ranked voxels in common over different training data sets when using the leave-one-subject out cross-validation, while activation maps have below 50% voxels in common between different selected subsets. This property of degree vs activation features is particularly important for interpretability of predictive modeling.

**2. Inter-hemispheric degree distributions.** A closer look at the degree distributions reveals that a large percentage of the differential connectivity appears to be due to long-distance, inter-hemispheric links. Figure 3a compares (normalized) histograms, for schizophrenic (red) versus normal (blue) groups, of the fraction of inter-hemispheric connections over the total number of connections, computed for each subject within the group. The schizophrenic group shows a significant bias towards low relative inter-hemispheric connectivity. A t-test analysis of the distributions indicates that differences are statistically significant (p=2.5x10-2). Moreover, it is evident that a major contributor to the high degree difference discussed before is the presence of a large number of inter-hemispheric connections in the normal group, which is lacking in schizophrenic group. Furthermore, we selected a bilateral regions of interest (ROI's) corresponding to left and right Brodmann Area 22 (roughly, the clusters in Figure 2a), *such that the linear activation for these ROI's was not significantly different between the groups*, even in the uncorrected case. For each subject, the link between the left and

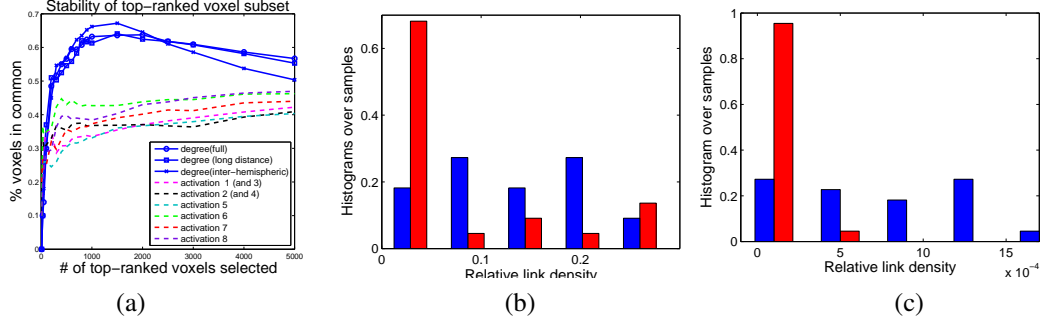

(a)                    (b)                  (c)

Figure 3: (a) Stability of feature subset selection over CV folds, i.e. the percent of voxels in common among the subsets of $k$ top variables selected at all CV folds. (b) Disruption of *global* inter-hemispheric connectivity. For each subject, we compute the fraction of inter-hemispheric connections over the total number of connections, and plot a normalized histogram over all subjects in a particular group (normal - blue, schizophrenic - red). (c) Disruption of *task-dependent* inter-hemispheric connectivity between specific ROIs (Brodmann Area 22 selected bilaterally). The ROIs were defined by a 9 mm radius ball centered at [x=-42, y=-24, z=3] and [x=42, y=-24, z=3].

| Feature | (GNB | SVM | MRF(0.01) |
|---|---|---|---|
| degree (D) | 27.5% | 27.5% | 27.5% |
| clustering coeff. (C) | 30.0% | 42.5% | 45.0% |
| geodesic dist. (G) | 67.5% | 45.0% | 45.0% |
| mean activation ($A$) | 40.0% | 45% | 72.5% |
| D + A | 27.5% | 27.5% | 32.5% |
| C + A | 27.5% | 45.0% | 55.0% |
| G + A | 45.0% | 45.0% | 72.5% |
| G +D +C | 37.5% | 27.5% | 27.5% |
| G+D+C+A | 30.0% | 27.5% | 32.5% |

(a)

| Feature | Error | False Pos | False Neg |
|---|---|---|---|
| degree (full) | **16%** | **27%** | **5%** |
| degree (long-distance) | **21%** | **32%** | **9%** |
| degree (inter-hemis) | 32% | 46% | 18% |
| activation 1 (and 3) | 54% | 29% | 82% |
| activation 2 (and 4) | 50% | 55% | 45% |
| activation 5 | 43% | 18% | 68% |
| activation 6 | 36% | 27% | 46% |
| activation 7 | 32% | 18% | 46% |
| activation 8 | 30% | 23% | 37% |

(b)

Table 1: Classification errors using (a) global features and (b) activation and degree maps (using SVM on the complete set of voxels (i.e., without voxel subset selection).

right ROIs was computed as the fraction of ROI-to-ROI connections over all connections; Figure 3c shows the normalized histograms. Clearly, the normal group displays a high density of inter-hemispheric connections, which are significantly disrupted in the schizophrenic group (p=3.7x10-7). This provides a strong indication that the group differences in connectivity cannot be explained by differences in local activation.

**3. Global features.** For each global feature (full list in Suppl. Mat.) we computed its mean for each group and p-value produced by the t-test, as well as the classification accuracies using our classifiers. While more details are presented in the supplemental material, we outline here the main observations: while mean activation (we used map 8, the best performer for SVM on the full set of voxels - see Table1b) had an relatively low p-value of $5.5 \times 10^{-4}$, as compared to less significant $p = 5.3 \times 10^{-2}$ for *mean-degree*, the predictive power of the latter, alone or in combination with some other features, was the best among global features reaching 27.5% in schizophrenic vs normal classification (Table 1a), while mean activation yielded more than 40% error with all classifiers.

**4. Classification results using degree vs. activation maps.** While mean-degree indicates the presence of discriminative information in voxel degrees, its generalization ability, though the best among global features and their combinations, is relatively poor. However, voxel-level degree maps turned out to be excellent predictive features, often outperforming activation features by far. Table 1b compares prediction made by SVM on complete maps (without voxel subset selection): both full and long-distance degree maps greatly outperform all activation maps, achieving 16% error vs. above 30% for even the best-performing activation map 8. Next, in Figure 4, we compare the predictive power of different maps when using all three classifiers: Support Vector Machines (SVM), Gaussian Naive Bayes (GNB) and sparse Gaussian Markov Random Field (MRF), on the subsets of $k$ top-ranked voxels, for a variety of $k$ values. We used the best-performing activation map 8 from the Table above, as well as maps 1 and 6 (that survived FDR); map 6 was also outperforming other activation maps in low-voxel regime. To avoid clutter, we only plot the two best-performing degree maps out of three (i.e., full and long-distance ones). For sparse MRF, we experimented with a variety of $\lambda$ values, ranging from 0.0001 to 10, and present the best results. We can see that: (**a**) *Degree maps frequently outperform activation maps*, for all classifiers we used; the differences are

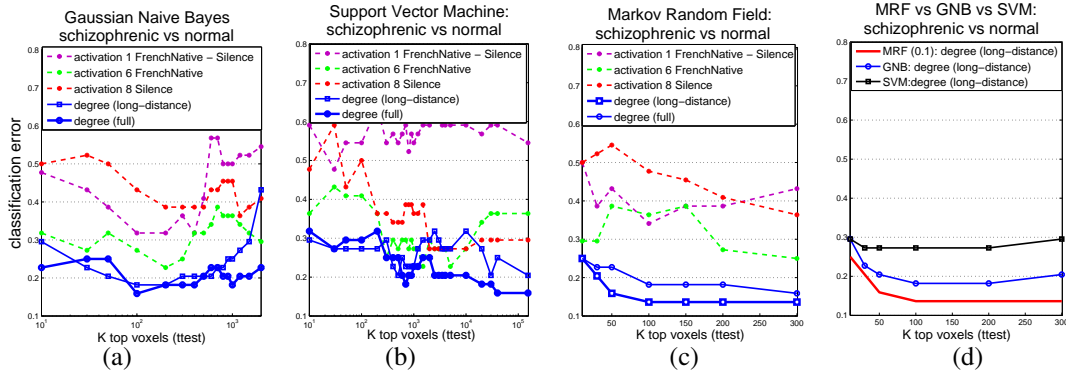

Figure 4: Classification results comparing (a) GNB, (b) SVM and (c) sparse MRF on degree versus activation contrast maps; (d) all three classifiers compared on long-distance degree maps (best-performing for MRF).

particularly noticeable when the number of selected voxels is relatively low. The most significant differences are observed for SVM in low-voxel (approx. $< 500$) and full-map regimes, as well as for MRF classifiers: it is remarkable that degree maps can achieve an impressively low error of 14% with only 100 most significant voxels, while even the best activation map 6 requires more than 200-300 to get just below 30% error; the other activation maps perform much worse, often above 30-40% error, or even just at the chance level. (**b**) Full and long-distance degree maps perform quite similarly, with long-distance map achieving the best result (14% error) using MRFs. (**c**) Among the activation maps only, while the map 8 ("Silence") outperforms others on the full set of voxels using SVM, its behavior in low-voxel regime is quite poor (always above 30-35% error); instead, map 6 ("FrenchNative") achieves best performance among activation maps in this regime[3]. (**d**) *MRF classifiers clearly outperform SVM and GNB*, possibly due to their ability to capture inter-voxel relationships that are highly discriminative between the two classes (see Figure 4d).

## 6   Summary

The contributions of this paper are two-fold. From a machine-learning and fMRI analysis perspective, we (a) introduced a novel feature-construction approach based on topological properties of functional networks, that is generally applicable to any multivariate-timeseries classification problems, and can outperform standard linear activation approaches in fMRI analysis field, (b) demonstrated advantages of this data-driven approach over prior-knowledge-based (ROI) approaches, and (c) demonstrated advantages of network-based classifiers (Markov Random Fields) over linear models (SVM, Naive Bayes) on fMRI data, suggesting to exploit voxel interactions in fMRI analyzes (i.e., treat brain as a network). From neuroscience perspective, we provided strong support for the hypothesis that schizophrenia is associated with the disruption of global, emergent brain properties which cannot be explained just by alteration of local activation patterns. Moreover, while prior art is mainly focused on exploring the differences between the functional and anatomical networks of schizophrenic patients versus healthy subjects [10, 1], this work, to our knowledge, is the first attempt to explore the generalization ability of predictive models of schizophrenia built on network features.

Finally, a word of caution. Note that the schizophrenia patients studied here have been selected for their prominent, persistent, and pharmaco-resistant auditory hallucinations [14], which might have increased their clinical homogeneity. However, the patient group is not representative of the full spectrum of the disease, and thus our conclusions may not necessarily apply to all schizophrenia patients, due to the clinical characteristics and size of the studied samples.

## Acknowledgements

We would like to thank Rahul Garg for his help with the data preprocessing and many stimulating discussions that contributed to the ideas of this paper, and Drs. André Galinowski, Thierry Gallarda, and Frank Bellivier who recruited and clinically rated the patients. We also would like to thank INSERM as promotor of the MR data acquired (project RBM $01 - 26$).

## Footnotes

[1]This is often referred to as the *disconnection hypothesis* [5, 15], and can be traced back to the early research on schizophrenia: in 1906, Wernicke [18] was the first one to postulate that anatomical disruption of association fiber tracts is at the roots of psychosis; in fact, the term schizophrenia was introduced by Bleuler [3] in 1911, and was meant to describe the separation (splitting) of different mental functions.

[2]Note that the inverse of the *empirical* covariance matrix, even if it exists, does not typically contain exact zeros. Therefore, an explicit sparsity constraint is usually added to the estimation process.

[3]We also observed that performing normalization really helped activation maps, since otherwise their performance could get much worse, especially with MRFs - we provide those results in supplemental material.

# References

[1] D.S. Bassett, E.T. Bullmore, B.A. Verchinski, V.S. Mattay, D.R. Weinberger, and A. Meyer-Lindenberg. Hierarchical organization of human cortical networks in health and schizophrenia. *J Neuroscience*, 28(37):9239–9248, 2008.

[2] A. Battle, G. Chechik, and D. Koller. Temporal and cross-subject probabilistic models for fmri prediction tasks. In B. Schölkopf, J. Platt, and T. Hoffman, editors, *Advances in Neural Information Processing Systems 19*, pages 121–128. MIT Press, Cambridge, MA, 2007.

[3] E. Bleuler. *Dementia Praecox or the Group of Schizophrenias*. International Universities Press, New York, NY, 1911.

[4] V.M. Eguiluz, D.R. Chialvo, G.A. Cecchi, M. Baliki, and A.V. Apkarian. Scale-free functional brain networks. *Physical Review Letters*, 94(018102), 2005.

[5] K.J. Friston and C.D. Frith. Schizophrenia: A Disconnection Syndrome? *Clinical Neuroscience*, (3):89–97, 1995.

[6] K.J. Friston, L. Harrison, and W.D. Penny. Dynamic Causal Modelling. *Neuroimage*, 19(4):1273–1302, Aug 2003.

[7] A.G. Garrity, G. D. Pearlson, K. McKiernan, D. Lloyd, K.A. Kiehl, and V.D. Calhoun. Aberrant "Default Mode" Functional Connectivity in Schizophrenia. *Am J Psychiatry*, 164:450–457, March 2007.

[8] J.V. Haxby, M.I. Gobbini, M.L. Furey, A.Ishai, J.L. Schouten, and P. Pietrini. Distributed and Overlapping Representations of Faces and Objects in Ventral Temporal Cortex. *Science*, 293(5539):2425–2430, 2001.

[9] K. J. Friston et al. Statistical parametric maps in functional imaging - a general linear approach. *Human Brain Mapping*, 2:189–210, 1995.

[10] Y. Liu, M. Liang, Y. Zhou, Y. He, Y. Hao, M. Song, C. Yu, H. Liu, Z. Liu, and T. Jiang. Disrupted Small-World Networks in Schizophrenia. *Brain*, 131:945–961, February 2008.

[11] T.M. Mitchell, R. Hutchinson, R.S. Niculescu, F. Pereira, X. Wang, M. Just, and S. Newman. Learning to Decode Cognitive States from Brain Images. *Machine Learning*, 57:145–175, 2004.

[12] O.Banerjee, L. El Ghaoui, and A. d'Aspremont. Model selection through sparse maximum likelihood estimation for multivariate gaussian or binary data. *Journal of Machine Learning Research*, 9:485–516, March 2008.

[13] F. Pereira and G. Gordon. The Support Vector Decomposition Machine. In *ICML2006*, pages 689–696, 2006.

[14] M. Plaze, D. Bartrs-Faz, JL Martinot, D. Januel, F. Bellivier, R. De Beaurepaire, S. Chanraud, J. Andoh, JP Lefaucheur, E. Artiges, C. Pallier, and ML Paillere-Martinot. Left superior temporal gyrus activation during sentence perception negatively correlates with auditory hallucination severity in schizophrenia patients. *Schizophrenia Research*, 87(1-3):109–115, 2006.

[15] K.E. Stephan, K.J. Friston, and C.D. Frith. Dysconnection in Schizophrenia: From Abnormal Synaptic Plasticity to Failures of Self-monitoring. *Schizophrenia Bulletin*, 35(3):509–527, 2009.

[16] A. J. Storkey, E. Simonotto, H. Whalley, S. Lawrie, L. Murray, and D. McGonigle. Learning structural equation models for fmri. In *Advances in Neural Information Processing Systems 19*, pages 1329–1336. 2007.

[17] B. Thirion, G. Flandin, P. Pinel, A. Roche, P. Ciuciu, and J.-B. Poline. Dealing with the shortcomings of spatial normalization: Multi-subject parcellation of fmri datasets. *Human Brain Mapping*, 27(8):678–693, 2006.

[18] C. Wernicke. Grundrisse der psychiatrie. *Thieme*, 1906.

[19] L. Zhang, D. Samaras, N. Alia-Klein, N. Volkow, and R. Goldstein. Modeling neuronal interactivity using dynamic bayesian networks. In *Advances in Neural Information Processing Systems 18*, pages 1593–1600. 2006.

